# $L_1$-Penalized Robust Estimation for a Class of Inverse Problems Arising in Multiview Geometry

**Arnak S. Dalalyan and Renaud Keriven**
IMAGINE/LabIGM,
Université Paris Est - Ecole des Ponts ParisTech,
Marne-la-Vallée, France
`dalalyan,keriven@imagine.enpc.fr`

## Abstract

We propose a new approach to the problem of robust estimation in multiview geometry. Inspired by recent advances in the sparse recovery problem of statistics, we define our estimator as a Bayesian maximum a posteriori with multivariate Laplace prior on the vector describing the outliers. This leads to an estimator in which the fidelity to the data is measured by the $L_\infty$-norm while the regularization is done by the $L_1$-norm. The proposed procedure is fairly fast since the outlier removal is done by solving one linear program (LP). An important difference compared to existing algorithms is that for our estimator it is not necessary to specify neither the number nor the proportion of the outliers. We present strong theoretical results assessing the accuracy of our procedure, as well as a numerical example illustrating its efficiency on real data.

## 1 Introduction

In the present paper, we are concerned with a class of non-linear inverse problems appearing in the structure and motion problem of multiview geometry. This problem, that have received a great deal of attention by the computer vision community in last decade, consists in recovering a set of 3D points (structure) and a set of camera matrices (motion), when only 2D images of the aforementioned 3D points by some cameras are available. Throughout this work we assume that the internal parameters of cameras as well as their orientations are known. Thus, only the locations of camera centers and 3D points are to be estimated. In solving the structure and motion problem by state-of-the-art methods, it is customary to start by establishing correspondences between pairs of 2D data points. We will assume in the present study that these point correspondences have been already established.

One can think of the structure and motion problem as the inverse problem of inverting the operator $\mathcal{O}$ that takes as input the set of 3D points and the set of cameras, and produces as output the 2D images of the 3D points by the cameras. This approach will be further formalized in the next section. Generally, the operator $\mathcal{O}$ is not injective, but in many situations (for example, when for each pair of cameras there are at least five 3D points in general position that are seen by these cameras [23]), there is only a small number of inputs, up to an overall similarity transform, having the same image by $\mathcal{O}$. In such cases, the solutions to the structure and motion problem can be found using algebraic arguments.

The main flaw of algebraic solutions is their sensitivity to the noise in the data: very often, thanks to the noise in the measurements, there is no input that could have generated the observed output. A natural approach to cope with such situations consists in searching for the input providing the closest possible output to the observed data. Then, a major issue is how to choose the metric in the output space. A standard approach [16] consists in measuring the distance between two elements

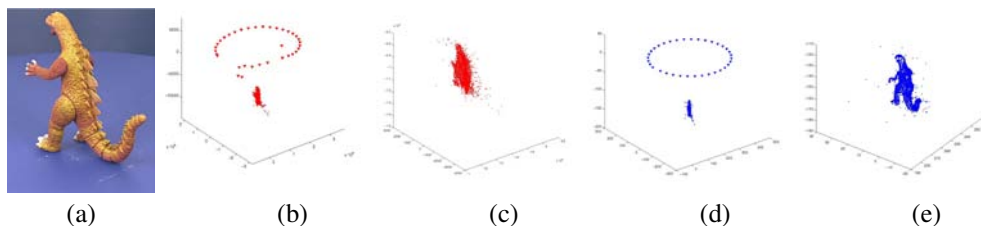

| (a) | (b) | (c) | (d) | (e) |

Figure 1: *(a) One image from the dinosaur sequence. Camera locations and scene points estimated by the blind $L_\infty$-cost minimization (b,c) and by the proposed "outlier aware" procedure (d,e).*

of the output space in the Euclidean $L_2$-norm. In the structure and motion problem with more than two cameras, this leads to a hard non-convex optimization problem. A particularly elegant way of circumventing the non-convexity issues inherent to the use of $L_2$-norm consists in replacing it by the $L_\infty$-norm [15, 18, 24, 25, 27, 13, 26]. It has been shown that, for a number of problems, $L_\infty$-norm based estimators can be computed very efficiently using, for example, the iterative bisection method [18, Algorithm 1, p. 1608] that solves a convex program at each iteration. There is however an issue with the $L_\infty$-techniques that dampens the enthusiasm of practitioners: it is highly sensitive to outliers (*c.f.* Fig. 1). In fact, among all $L_q$-metrics with $q \geq 1$, the $L_\infty$-metric is the most seriously affected by the outliers in the data. Two procedures have been introduced [27, 19] that make the $L_\infty$-estimator less sensitive to outliers. Although these procedures demonstrate satisfactory empirical performance, they suffer from a lack of sufficient theoretical support assessing the accuracy of produced estimates.

The purpose of the present work is to introduce and to theoretically investigate a new procedure of estimation in presence of noise and outliers. Our procedure combines $L_\infty$-norm for measuring the fidelity to the data and $L_1$-norm for regularization. It can be seen as a maximum a posteriori (MAP) estimator under uniformly distributed random noise and a sparsity favoring prior on the vector of outliers. Interestingly, this study bridges the work on the robust estimation in multiview geometry [12, 27, 19, 21] and the theory of sparse recovery in statistics and signal processing [10, 2, 5, 6].

The rest of the paper is organized as follows. The next section gives the precise formulation of the translation estimation and triangulation problem to which the presented methodology can be applied. A brief review of the $L_\infty$-norm minimization algorithm is presented in Section 3. In Section 4, we introduce the statistical framework and derive a new procedure as a MAP estimator. The main result on the accuracy of this procedure is stated and proved in Section 5, while Section 6 contains some numerical experiments. The methodology of our study is summarized in Section 7.

## 2  Translation estimation and triangulation

Let us start by presenting a problem of multiview geometry to which our approach can be successfully applied, namely the problem of translation estimation and triangulation in the case of known rotations. For rotation estimation algorithms, we refer the interested reader to [22, 14] and the references therein.

Let $P_i^*$, $i = 1, \dots, m$, be a sequence of $m$ cameras that are known up to a translation. Recall that a camera is characterized by a $3 \times 4$ matrix $P$ with real entries that can be written as $P = K[R|\mathbf{t}]$, where $K$ is an invertible $3 \times 3$ matrix called the camera calibration matrix, $R$ is a $3 \times 3$ rotation matrix and $\mathbf{t} \in \mathbb{R}^3$. We will refer to $\mathbf{t}$ as the translation of the camera $P$. We can thus write $P_i^* = K_i[R_i|\mathbf{t}_i^*]$, $i = 1, \dots, m$. For a set of unknown scene points $\mathbf{U}_j^*$, $j = 1, \dots, n$, expressed in homogeneous coordinates (*i.e.*, $\mathbf{U}_j^*$ is an element of the projective space $\mathbb{P}^3$), we assume that noisy images of each $\mathbf{U}_j^*$ by some cameras $P_i^*$ are observed. Thus, we have at our disposal the measurements

$$\mathbf{x}_{ij} = \frac{1}{\mathbf{e}_3^\mathsf{T} P_i^* \mathbf{U}_j^*} \begin{bmatrix} \mathbf{e}_1^\mathsf{T} P_i^* \mathbf{U}_j^* \\ \mathbf{e}_2^\mathsf{T} P_i^* \mathbf{U}_j^* \end{bmatrix} + \boldsymbol{\xi}_{ij}, \quad \begin{matrix} j = 1, \dots, n, \\ i \in I_j, \end{matrix} \tag{1}$$

where $\mathbf{e}_\ell$, $\ell = 1, 2, 3$, stands for the unit vector of $\mathbb{R}^3$ having one as the $\ell^{\text{th}}$ coordinate and $I_j$ is the set of indices of cameras for which the point $\mathbf{U}_j^*$ is visible. We assume that the set $\{\mathbf{U}_j^*\}$ does not contain points at infinity: $\mathbf{U}_j^* = [\mathbf{X}_j^{*\mathsf{T}}|1]^\mathsf{T}$ for some $\mathbf{X}_j^* \in \mathbb{R}^3$ and for every $j = 1, \dots, n$.

We are now in a position to state the problem of translation estimation and triangulation in the context of multiview geometry. It consists in recovering the 3-vectors $\{\mathbf{t}_i^*\}$ (translation estimation) and the 3D points $\{\mathbf{X}_j^*\}$ (triangulation) from the noisy measurements $\{\mathbf{x}_{ij}; j = 1, \ldots, n; i \in I_j\} \subset \mathbb{R}^2$. In what follows, we use the notation $\boldsymbol{\theta}^* = (\mathbf{t}_1^{*\mathsf{T}}, \ldots, \mathbf{t}_m^{*\mathsf{T}}, \mathbf{X}_1^{*\mathsf{T}}, \ldots, \mathbf{X}_n^{*\mathsf{T}})^{\mathsf{T}} \in \mathbb{R}^{3(m+n)}$. Thus, we are interested in estimating $\boldsymbol{\theta}^*$.

**Remark 1** (Cheirality). *It should be noted right away that if the point $\mathbf{U}_j^*$ is in front of the camera $\mathbf{P}_i^*$, then $\mathbf{e}_3^{\mathsf{T}} \mathbf{P}_i^* \mathbf{U}_j^* \geq 0$. This is termed cheirality condition. Furthermore, we will assume that none of the true 3D points $\mathbf{U}_j^*$ lies on the principal plane of a camera $\mathbf{P}_i^*$. This assumption implies that $\mathbf{e}_3^{\mathsf{T}} \mathbf{P}_i^* \mathbf{U}_j^* > 0$ so that the quotients $\mathbf{e}_\ell^{\mathsf{T}} \mathbf{P}_i^* \mathbf{U}_j^* / \mathbf{e}_3^{\mathsf{T}} \mathbf{P}_i^* \mathbf{U}_j^*$, $\ell = 1, 2$, are well defined.*

**Remark 2** (Identifiability). *The parameter $\boldsymbol{\theta}$ we have just defined is, in general, not identifiable from the measurements $\{\mathbf{x}_{ij}\}$. In fact, one easily checks that, for every $\alpha \neq 0$ and for every $\mathbf{t} \in \mathbb{R}^3$, the parameters $\{\mathbf{t}_i^*, \mathbf{X}_j^*\}$ and $\{\alpha(\mathbf{t}_i^* - \mathbf{R}_i \mathbf{t}), \alpha(\mathbf{X}_j^* + \mathbf{t})\}$ generate the same measurements. To cope with this issue, we assume that $\mathbf{t}_1^* = \mathbf{0}_3$ and that $\min_{i,j} \mathbf{e}_3^{\mathsf{T}} \mathbf{P}_i^* \mathbf{U}_j^* = 1$. Thus, in what follows we assume that $\mathbf{t}_1^*$ is removed from $\boldsymbol{\theta}^*$ and $\boldsymbol{\theta}^* \in \mathbb{R}^{3(m+n-1)}$. Further assumptions ensuring the identifiability of $\boldsymbol{\theta}^*$ are given below.*

## 3 Estimation by Sequential Convex Programming

This section presents results on the estimation of $\boldsymbol{\theta}$ based on the reprojection error (RE) minimization. This material is essential for understanding the results that are at the core of the present work. In what follows, for every $s \geq 1$, we denote by $\|\mathbf{x}\|_s$ the $L_s$-norm of a vector $\mathbf{x}$, *i.e.* $\|\mathbf{x}\|_s^s = \sum_j |x_j|^s$ if $\mathbf{x} = (x_1, \ldots, x_d)^{\mathsf{T}}$. As usual, we extend this to $s = +\infty$ by setting $\|\mathbf{x}\|_\infty = \max_j |x_j|$.

A classical method [16] for estimating the parameter $\boldsymbol{\theta}$ is based on minimizing the sum of the squared REs. This defines the estimator $\widehat{\boldsymbol{\theta}}$ as a minimizer of the cost function $\mathcal{C}_{2,2}(\boldsymbol{\theta}) = \sum_{i,j} \|\mathbf{x}_{ij} - \mathbf{x}_{ij}(\boldsymbol{\theta})\|_2^2$, where $\mathbf{x}_{ij}(\boldsymbol{\theta}) := \left[\mathbf{e}_1^{\mathsf{T}} \mathbf{P}_i \mathbf{U}_j; \mathbf{e}_2^{\mathsf{T}} \mathbf{P}_i \mathbf{U}_j\right]^{\mathsf{T}} / \mathbf{e}_3^{\mathsf{T}} \mathbf{P}_i \mathbf{U}_j$ is the 2-vector that we would obtain if $\boldsymbol{\theta}$ were the true parameter. It can also be written as

$$\mathbf{x}_{ij}(\boldsymbol{\theta}) = \left[\frac{\mathbf{e}_1^{\mathsf{T}} \mathbf{K}_i(\mathbf{R}_i \mathbf{X}_j + \mathbf{t}_i)}{\mathbf{e}_3^{\mathsf{T}} \mathbf{K}_i(\mathbf{R}_i \mathbf{X}_j + \mathbf{t}_i)}; \frac{\mathbf{e}_2^{\mathsf{T}} \mathbf{K}_i(\mathbf{R}_i \mathbf{X}_j + \mathbf{t}_i)}{\mathbf{e}_3^{\mathsf{T}} \mathbf{K}_i(\mathbf{R}_i \mathbf{X}_j + \mathbf{t}_i)}\right]^{\mathsf{T}}. \tag{2}$$

The minimization of $\mathcal{C}_{2,2}$ is a hard nonconvex problem. In general, it does not admit closed-form solution and the existing iterative algorithms may often get stuck in local minima. An ingenious idea to overcome this difficulty [15, 17] is based on the minimization of the $L_\infty$ cost function

$$\mathcal{C}_{\infty,s}(\boldsymbol{\theta}) = \max_{j=1,\ldots,n} \max_{i \in I_j} \|\mathbf{x}_{ij} - \mathbf{x}_{ij}(\boldsymbol{\theta})\|_s, \qquad s \in [1, +\infty]. \tag{3}$$

Note that the substitution of the $L_2$-cost function by the $L_\infty$-cost function has been proved to lead to improved algorithms in other estimation problems as well, cf., *e.g.*, [8]. This cost function has a clear practical advantage in that all its sublevel sets are convex. This property ensures that all minima of $\mathcal{C}_{\infty,s}$ form a convex set and that an element of this set can be computed by solving a sequence of convex programs [18], *e.g.*, by the bisection algorithm. Note that for $s = 1$ and $s = +\infty$, the minimization of $\mathcal{C}_{\infty,s}$ can be recast in a sequence of LPs. The main idea behind the bisection algorithm can be summarized as follows. We aim to designate an algorithm computing $\widehat{\boldsymbol{\theta}}_s \in \arg\min_{\boldsymbol{\theta}} \mathcal{C}_{\infty,s}(\boldsymbol{\theta})$, for any prespecified $s \geq 1$, over the set of all vectors $\boldsymbol{\theta}$ satisfying the cheirality condition. Let us introduce the residuals $\mathbf{r}_{ij}(\boldsymbol{\theta}) = \mathbf{x}_{ij} - \mathbf{x}_{ij}(\boldsymbol{\theta})$ that can be represented as

$$\mathbf{r}_{ij}(\boldsymbol{\theta}) = \left[\frac{\mathbf{a}_{ij1}^{\mathsf{T}} \boldsymbol{\theta}}{\mathbf{c}_{ij}^{\mathsf{T}} \boldsymbol{\theta}}; \frac{\mathbf{a}_{ij2}^{\mathsf{T}} \boldsymbol{\theta}}{\mathbf{c}_{ij}^{\mathsf{T}} \boldsymbol{\theta}}\right]^{\mathsf{T}}, \tag{4}$$

for some vectors $\mathbf{a}_{ij\ell}, \mathbf{c}_{ij} \in \mathbb{R}^2$. Furthermore, as presented in Remark 2, the cheirality conditions imply the set of linear constraints $\mathbf{c}_{ij}^{\mathsf{T}} \boldsymbol{\theta} \geq 1$. Thus, the problem of computing $\widehat{\boldsymbol{\theta}}_s$ can be rewritten as

$$\text{minimize} \quad \gamma \quad \text{subject to} \quad \begin{cases} \|\mathbf{r}_{ij}(\boldsymbol{\theta})\|_s \leq \gamma, \\ \mathbf{c}_{ij}^{\mathsf{T}} \boldsymbol{\theta} \geq 1. \end{cases} \tag{5}$$

Note that the inequality $\|\mathbf{r}_{ij}(\boldsymbol{\theta})\|_s \leq \gamma$ can be replaced by $\|\mathbf{A}_{ij}^{\mathsf{T}} \boldsymbol{\theta}\|_s \leq \gamma \mathbf{c}_{ij}^{\mathsf{T}} \boldsymbol{\theta}$ with $\mathbf{A}_{ij} = [\mathbf{a}_{ij1}; \mathbf{a}_{ij2}]$. Although (5) is not a convex problem, its solution can be well approximated by solving a sequence of convex feasibility problems.

## 4 Robust estimation by linear programming

This and the next sections contain the main theoretical contribution of the present work. We start with the precise formulation of the statistical model. We then exhibit a prior distribution on the unknown parameters of the model that leads to a MAP estimator.

### 4.1 The statistical model

Let us first observe that, in view of (1) and (4), the model we are considering can be rewritten as

$$\left[\frac{\mathbf{a}_{ij1}^{\mathsf{T}}\boldsymbol{\theta}^*}{\mathbf{c}_{ij}^{\mathsf{T}}\boldsymbol{\theta}^*}; \frac{\mathbf{a}_{ij2}^{\mathsf{T}}\boldsymbol{\theta}^*}{\mathbf{c}_{ij}^{\mathsf{T}}\boldsymbol{\theta}^*}\right]^{\mathsf{T}} = \boldsymbol{\xi}_{ij}, \quad j = 1, \ldots, n; \ i \in I_j. \tag{6}$$

Let $N = 2\sum_{j=1}^n I_j$ be the total number of measurements and let $M = 3(n + m - 1)$ be the size of the vector $\boldsymbol{\theta}^*$. Let us denote by $\mathtt{A}$ (resp. $\mathtt{C}$) the $M \times N$ matrix formed by the concatenation of the column-vectors $\mathbf{a}_{ij\ell}$ (resp. $\mathbf{c}_{ij}$[1]). Similarly, let us denote by $\boldsymbol{\xi}$ the $N$-vector formed by concatenating the vectors $\boldsymbol{\xi}_{ij}$. In these notation, Eq. (6) is equivalent to $\mathbf{a}_p^{\mathsf{T}}\boldsymbol{\theta}^* = (\mathbf{c}_p^{\mathsf{T}}\boldsymbol{\theta}^*)\boldsymbol{\xi}_p$, $p = 1, \ldots, N$. This equation defines the statistical model in the case where there is no outlier. To extend this model to cover the situation where some outliers are present in the measurements, we introduce the vector $\boldsymbol{\omega}^* \in \mathbb{R}^N$ defined by $\omega_p^* = \mathbf{a}_p^{\mathsf{T}}\boldsymbol{\theta}^* - (\mathbf{c}_p^{\mathsf{T}}\boldsymbol{\theta}^*)\boldsymbol{\xi}_p$ so that $\omega_p^* = 0$ if the $p^{\text{th}}$ measurement is an inlier and $|\omega_p^*| > 0$ otherwise. This leads us to the model:

$$\mathtt{A}^{\mathsf{T}}\boldsymbol{\theta}^* = \boldsymbol{\omega}^* + \mathrm{diag}(\mathtt{C}^{\mathsf{T}}\boldsymbol{\theta}^*)\boldsymbol{\xi}, \tag{7}$$

where $\mathrm{diag}(\mathbf{v})$ stands for the diagonal matrix having the components of $\mathbf{v}$ as diagonal entries.

**Statement of the problem:** *Given the matrices $\mathtt{A}$ and $\mathtt{C}$, estimate the parameter-vector $\boldsymbol{\beta}^* = [\boldsymbol{\theta}^{*\mathsf{T}}; \boldsymbol{\omega}^{*\mathsf{T}}]^{\mathsf{T}}$ based on the following prior information:*
  $\mathrm{C}_1$: *Eq. (7) holds with some small noise vector $\boldsymbol{\xi}$,*
  $\mathrm{C}_2$: $\min_p \mathbf{c}_p^{\mathsf{T}}\boldsymbol{\theta}^* = 1$,
  $\mathrm{C}_3$: $\boldsymbol{\omega}^*$ *is sparse,* i.e., *only a small number of coordinates of $\boldsymbol{\omega}^*$ are different from zero.*

### 4.2 Sparsity prior and MAP estimator

To derive an estimator of the parameter $\boldsymbol{\beta}^*$, we place ourselves in the Bayesian framework. To this end, we impose a probabilistic structure on the noise vector $\boldsymbol{\xi}$ and introduce a prior distribution on the unknown vector $\boldsymbol{\beta}$.

Since the noise $\boldsymbol{\xi}$ represents the difference (in pixels) between the measurements and the true image points, it is naturally bounded and, generally, does not exceeds the level of a few pixels. Therefore, it is reasonable to assume that the components of $\boldsymbol{\xi}$ are uniformly distributed in some compact set of $\mathbb{R}^2$, centered at the origin. We assume in what follows that the subvectors $\xi_{ij}$ of $\boldsymbol{\xi}$ are uniformly distributed in the square $[-\sigma, \sigma]^2$ and are mutually independent. Note that this implies that all the coordinates of $\boldsymbol{\xi}$ are independent. In practice, this assumption can be enforced by decorrelating the measurements using the empirical covariance matrix [20]. We define the prior on $\boldsymbol{\theta}$ as the uniform distribution on the polytope $\mathcal{P} = \{\boldsymbol{\theta} \in \mathbb{R}^M : \mathtt{C}^{\mathsf{T}}\boldsymbol{\theta} \geq 1\}$, where the inequality is understood componentwise. The density of this distribution is $p_1(\boldsymbol{\theta}) \propto \mathbf{1}_{\mathcal{P}}(\boldsymbol{\theta})$, where $\propto$ stands for the proportionality relation and $\mathbf{1}_{\mathcal{P}}(\boldsymbol{\theta}) = 1$ if $\boldsymbol{\theta} \in \mathcal{P}$ and 0 otherwise. When $\mathcal{P}$ is unbounded, this results in an improper prior, which is however not a problem for defining the Bayes estimator.

The task of choosing a prior on $\boldsymbol{\omega}$ is more delicate in that it should reflect the information that $\boldsymbol{\omega}$ is sparse. The most natural prior would be the one having a density which is a decreasing function of the $L_0$-norm of $\boldsymbol{\omega}$, *i.e.*, of the number of its nonzero coefficients. However, the computation of estimators based on this type of priors is NP-hard. An approach for overcoming this difficulty relies on using the $L_1$-norm instead of the $L_0$-norm. Following this idea, we define the prior distribution on $\boldsymbol{\omega}$ by the probability density $p_2(\boldsymbol{\omega}) \propto f(\|\boldsymbol{\omega}\|_1)$, where $f$ is some decreasing function[2] defined on $[0, \infty)$. Assuming in addition that $\boldsymbol{\theta}$ and $\boldsymbol{\omega}$ are independent, we get the following prior on $\boldsymbol{\beta}$:

$$\pi(\boldsymbol{\beta}) = \pi(\boldsymbol{\theta}; \boldsymbol{\omega}) \propto \mathbf{1}_{\mathcal{P}}(\boldsymbol{\theta}) \cdot f(\|\boldsymbol{\omega}\|_1). \tag{8}$$

**Theorem 1.** *Assume that the noise $\boldsymbol{\xi}$ has independent entries which are uniformly distributed in $[-\sigma, \sigma]$ for some $\sigma > 0$, then the MAP estimator $\widehat{\boldsymbol{\beta}} = [\widehat{\boldsymbol{\theta}}^\mathsf{T}; \widehat{\boldsymbol{\omega}}^\mathsf{T}]^\mathsf{T}$ based on the prior $\pi$ defined by Eq. (8) is the solution of the optimization problem:*

$$\text{minimize} \qquad \|\boldsymbol{\omega}\|_1 \qquad \text{subject to} \qquad \begin{cases} |\mathbf{a}_p^\mathsf{T}\boldsymbol{\theta} - \omega_p| \leq \sigma\mathbf{c}_p^\mathsf{T}\boldsymbol{\theta}, \ \forall p \\ \mathbf{c}_p^\mathsf{T}\boldsymbol{\theta} \geq 1, \ \forall p. \end{cases} \tag{9}$$

The proof of this theorem is a simple exercise and is left to the reader.

**Remark 3** (Condition $C_2$). *One easily checks that any solution of (9) satisfies condition $C_2$. Indeed, if for some solution $\widehat{\boldsymbol{\beta}}$ it were not the case, then $\tilde{\boldsymbol{\beta}} = \widehat{\boldsymbol{\beta}}/\min_p \mathbf{c}_p^\mathsf{T}\boldsymbol{\theta}$ would satisfy the constraints of (9) and $\tilde{\boldsymbol{\omega}}$ would have a smaller $L_1$-norm than that of $\widehat{\boldsymbol{\omega}}$, which is in contradiction with the fact that $\widehat{\boldsymbol{\beta}}$ solves (9).*

**Remark 4** (The role of $\sigma$). *In the definition of $\widehat{\boldsymbol{\beta}}$, $\sigma$ is a free parameter that can be interpreted as the level of separation of inliers from outliers. The proposed algorithm implicitly assumes that all the measurements $\mathbf{x}_{ij}$ for which $\|\boldsymbol{\xi}_{ij}\|_\infty > \sigma$ are outliers, while all the others are treated as inliers.*

If $\sigma$ is unknown, a reasonable way of acting is to impose a prior distribution on the possible values of $\sigma$ and to define the estimator $\widehat{\boldsymbol{\beta}}$ as a MAP estimator based on the prior incorporating the uncertainty on $\sigma$. When there are no outliers and the prior on $\sigma$ is decreasing, this approach leads to the estimator minimizing the $L_\infty$ cost function. In the presence of outliers, the shape of the prior on $\sigma$ becomes more important for the definition of the estimator. This is an interesting point for future investigation.

### 4.3 Two-step procedure

Building on the previous arguments, we introduce the following two-step algorithm.

> **Input:** $\{\mathbf{a}_p, \mathbf{c}_p; p = 1, \ldots, N\}$ and $\sigma$.
> **Step 1:** Compute $[\widehat{\boldsymbol{\theta}}^\mathsf{T}; \widehat{\boldsymbol{\omega}}^\mathsf{T}]^\mathsf{T}$ as a solution to (9) and set $J = \{p : \widehat{\omega}_p = 0\}$.
> **Step 2:** Apply the bisection algorithm to the reduced data set $\{\mathbf{x}_p; p \in J\}$.

Two observations are in order. First, when applying the bisection algorithm at Step 2, we can use $\mathcal{C}_{\infty,s}(\widehat{\boldsymbol{\theta}})$ as the initial value of $\gamma_u$. The second observation is that a better way of acting would be to minimize the weighted $L_1$-norm of $\boldsymbol{\omega}$, where the weight assigned to $\omega_p$ is inversely proportional to the depth $\mathbf{c}_p^\mathsf{T}\boldsymbol{\theta}^*$. Since $\boldsymbol{\theta}^*$ is unknown, a reasonable strategy consists in adding a step in between Step 1 and Step 2, which performs the weighted minimization with weights $\{(\mathbf{c}_p^\mathsf{T}\widehat{\boldsymbol{\theta}})^{-1}; p = 1, \ldots, N\}$.

## 5 Accuracy of estimation

Let us introduce some additional notation. Recall the definition of $\mathcal{P}$ and set $\partial\mathcal{P} = \{\boldsymbol{\theta} : \min_p \mathbf{c}_p^T\boldsymbol{\theta} = 1\}$ and $\Delta\mathcal{P}^* = \{\boldsymbol{\theta} - \boldsymbol{\theta}' : \boldsymbol{\theta}, \boldsymbol{\theta}' \in \partial\mathcal{P}, \boldsymbol{\theta} \neq \boldsymbol{\theta}'\}$. For every subset of indices $J \subset \{1, \ldots, N\}$, we denote by $\mathtt{A}_J$ the $M \times N$ matrix obtained from $\mathtt{A}$ by replacing the columns that have an index outside $J$ by zero. Furthermore, let us define

$$\delta_J(\boldsymbol{\theta}) = \sup_{\boldsymbol{\theta}' \in \partial\mathcal{P}, \mathtt{A}^\mathsf{T}\boldsymbol{\theta}' \neq \mathtt{A}^\mathsf{T}\boldsymbol{\theta}} \frac{\|\mathtt{A}_J^\mathsf{T}(\boldsymbol{\theta}' - \boldsymbol{\theta})\|_2}{\|\mathtt{A}^\mathsf{T}(\boldsymbol{\theta}' - \boldsymbol{\theta})\|_2}, \qquad \forall J \subset \{1, \ldots, N\}, \quad \forall \boldsymbol{\theta} \in \partial\mathcal{P}. \tag{10}$$

One easily checks that $\delta_J \in [0, 1]$ and $\delta_J \leq \delta_{J'}$ if $J \subset J'$.

**Assumption A:** *The real number $\lambda$ defined by $\lambda = \min_{\mathbf{g} \in \Delta\mathcal{P}^*} \|\mathtt{A}^\mathsf{T}\mathbf{g}\|_2/\|\mathbf{g}\|_2$ is strictly positive.*

Assumption A is necessary for identifying the parameter vector $\boldsymbol{\theta}^*$ even in the case without outliers. In fact, if $\boldsymbol{\omega}^* = \mathbf{0}$, and if Assumption A is not fulfilled, then[3] $\exists \mathbf{g} \in \Delta\mathcal{P}^*$ such that $\mathtt{A}^\mathsf{T}\mathbf{g} = \mathbf{0}$. That is, given the matrices $\mathtt{A}$ and $\mathtt{C}$, there are two distinct vectors $\boldsymbol{\theta}^1$ and $\boldsymbol{\theta}^2$ in $\partial\mathcal{P}$ such that $\mathtt{A}^\mathsf{T}\boldsymbol{\theta}^1 = \mathtt{A}^\mathsf{T}\boldsymbol{\theta}^2$. Therefore, if eventually $\boldsymbol{\theta}^1$ is the true parameter vector satisfying $C_1$ and $C_3$, then $\boldsymbol{\theta}^2$ satisfies these conditions as well. As a consequence, the true vector cannot be accurately estimated.

## 5.1 The noise free case

To evaluate the quality of estimation, we first place ourselves in the case where $\sigma = 0$. The estimator $\widehat{\boldsymbol{\beta}}$ of $\boldsymbol{\beta}^*$ is then defined as a solution to the optimization problem

$$\min \|\boldsymbol{\omega}\|_1 \quad \text{over } \boldsymbol{\beta} = \begin{bmatrix} \boldsymbol{\theta} \\ \boldsymbol{\omega} \end{bmatrix} \text{ s.t. } \begin{cases} \mathtt{A}^\mathsf{T}\boldsymbol{\theta} = \boldsymbol{\omega} \\ \mathtt{C}^\mathsf{T}\boldsymbol{\theta} \geq 1 \end{cases}. \tag{11}$$

From now on, for every index set $T$ and for every vector $\mathbf{h}$, $\mathbf{h}_T$ stands for the vector equal to $\mathbf{h}$ on an index set $T$ and zero elsewhere. The complementary set of $T$ will be denoted by $T^c$.

**Theorem 2.** *Let Assumption A be fulfilled and let $T_0$ (resp. $T_1$) denote the index set corresponding to the locations of $S$ largest entries[4] of $\boldsymbol{\omega}^*$ (resp. $(\boldsymbol{\omega}^* - \widehat{\boldsymbol{\omega}})_{T_0^c}$). If $\delta_{T_0}(\boldsymbol{\theta}^*) + \delta_{T_0 \cup T_1}(\boldsymbol{\theta}^*) < 1$ then, for some constant $C_0$, it holds:*

$$\|\widehat{\boldsymbol{\beta}} - \boldsymbol{\beta}^*\|_2 \leq C_0 \|\boldsymbol{\omega}^* - \boldsymbol{\omega}_S^*\|_1, \tag{12}$$

*where $\boldsymbol{\omega}_S^*$ stands for the vector $\boldsymbol{\omega}^*$ with all but the $S$-largest entries set to zero. In particular, if $\boldsymbol{\omega}^*$ has no more than $S$ nonzero entries, then the estimation is exact: $\widehat{\boldsymbol{\beta}} = \boldsymbol{\beta}^*$.*

*Proof.* We set $\mathbf{h} = \boldsymbol{\omega}^* - \widehat{\boldsymbol{\omega}}$ and $\mathbf{g} = \boldsymbol{\theta}^* - \widehat{\boldsymbol{\theta}}$. It follows from Remark 3 that $\mathbf{g} \in \Delta\mathcal{P}$. To proceed with the proof, we need the following auxiliary result, the proof of which can be easily deduced from [4].

**Lemma 1.** *Let $\mathbf{v} \in \mathbb{R}^d$ be some vector and let $S \leq d$ be a positive integer. If we denote by $T$ the indices of $S$ largest entries of the vector $|\mathbf{v}|$, then $\|\mathbf{v}_{T^c}\|_2 \leq S^{-1/2}\|\mathbf{v}\|_1$.*

Applying Lemma 1 to the vector $\mathbf{v} = \mathbf{h}_{T_0^c}$ and to the index set $T = T_1$, we get

$$\|\mathbf{h}_{(T_0 \cup T_1)^c}\|_2 \leq S^{-1/2}\|\mathbf{h}_{T_0^c}\|_1. \tag{13}$$

On the other hand, summing up the inequalities $\|\mathbf{h}_{T_0^c}\|_1 \leq \|(\boldsymbol{\omega}^* - \mathbf{h})_{T_0^c}\|_1 + \|\boldsymbol{\omega}_{T_0^c}^*\|_1$ and $\|\boldsymbol{\omega}_{T_0}^*\|_1 \leq \|(\boldsymbol{\omega}^* - \mathbf{h})_{T_0}\|_1 + \|\mathbf{h}_{T_0}\|_1$, and using the relation $\|(\boldsymbol{\omega}^* - \mathbf{h})_{T_0}\|_1 + \|(\boldsymbol{\omega}^* - \mathbf{h})_{T_0^c}\|_1 = \|\boldsymbol{\omega}^* - \mathbf{h}\|_1 = \|\widehat{\boldsymbol{\omega}}\|_1$, we get

$$\|\mathbf{h}_{T_0^c}\|_1 + \|\boldsymbol{\omega}_{T_0}^*\|_1 \leq \|\widehat{\boldsymbol{\omega}}\|_1 + \|\boldsymbol{\omega}_{T_0^c}^*\|_1 + \|\mathbf{h}_{T_0}\|_1. \tag{14}$$

Since $\boldsymbol{\beta}^*$ satisfies the constraints of the optimization problem (11) a solution of which is $\widehat{\boldsymbol{\beta}}$, we have $\|\widehat{\boldsymbol{\omega}}\|_1 \leq \|\boldsymbol{\omega}^*\|_1$. This inequality, in conjunction with (13) and (14), implies

$$\|\mathbf{h}_{(T_0 \cup T_1)^c}\|_2 \leq S^{-1/2}\|\mathbf{h}_{T_0}\|_1 + 2S^{-1/2}\|\boldsymbol{\omega}_{T_0^c}^*\|_1 \leq \|\mathbf{h}_{T_0}\|_2 + 2S^{-1/2}\|\boldsymbol{\omega}_{T_0^c}^*\|_1, \tag{15}$$

where the last step follows from the Cauchy-Schwartz inequality. Using once again the fact that both $\widehat{\boldsymbol{\beta}}$ and $\boldsymbol{\beta}^*$ satisfy the constraints of (11), we get $\mathbf{h} = \mathtt{A}^\mathsf{T}\mathbf{g}$. Therefore,

$$\begin{aligned}
\|\mathbf{h}\|_2 &\leq \|\mathbf{h}_{T_0 \cup T_1}\|_2 + \|\mathbf{h}_{(T_0 \cup T_1)^c}\|_2 \leq \|\mathbf{h}_{T_0 \cup T_1}\|_2 + \|\mathbf{h}_{T_0}\|_2 + 2S^{-1/2}\|\boldsymbol{\omega}_{T_0^c}^*\|_1 \\
&= \|\mathtt{A}_{T_0 \cup T_1}^\mathsf{T}\mathbf{g}\|_2 + \|\mathtt{A}_{T_0}^\mathsf{T}\mathbf{g}\|_2 + 2S^{-1/2}\|\boldsymbol{\omega}_{T_0^c}^*\|_1 \leq (\delta_{2S} + \delta_S)\|\mathtt{A}^\mathsf{T}\mathbf{g}\|_2 + 2S^{-1/2}\|\boldsymbol{\omega}_{T_0^c}^*\|_1 \\
&= (\delta_{2S} + \delta_S)\|\mathbf{h}\|_2 + 2S^{-1/2}\|\boldsymbol{\omega}_{T_0^c}^*\|_1. 
\end{aligned} \tag{16}$$

Since $\boldsymbol{\omega}_{T_0^c}^* = \boldsymbol{\omega}^* - \boldsymbol{\omega}_S$, the last inequality yields $\|\mathbf{h}\|_2 \leq (2S^{-1/2}/(1 - \delta_S - \delta_{2S}))\|\boldsymbol{\omega}^* - \boldsymbol{\omega}_S^*\|_1$. To complete the proof, it suffices to observe that

$$\|\widehat{\boldsymbol{\beta}} - \boldsymbol{\beta}^*\|_2 \leq \|\mathbf{g}\|_2 + \|\mathbf{h}\|_2 \leq \lambda^{-1}\|\mathtt{A}\mathbf{g}\|_2 + \|\mathbf{h}\|_2 = (\lambda^{-1} + 1)\|\mathbf{h}\|_2 \leq C_0\|\boldsymbol{\omega}^* - \boldsymbol{\omega}_S^*\|_1. \quad \square$$

**Remark 5.** *The assumption $\delta_{T_0}(\boldsymbol{\theta}^*) + \delta_{T_0 \cup T_1}(\boldsymbol{\theta}^*) < 1$ is close in spirit to the restricted isometry assumption (cf., e.g., [10, 6, 3] and the references therein). It is very likely that results similar to that of Theorem 2 hold under other kind of assumptions recently introduced in the theory of $L_1$-minimization [11, 29, 2]. This investigation is left for future research.*

We emphasize that the constant $C_0$ is rather small. For example, if $\delta_{T_0}(\boldsymbol{\theta}^*) + \delta_{T_0 \cup T_1}(\boldsymbol{\theta}^*) = 0.5$, then $\max(\|\widehat{\boldsymbol{\omega}} - \boldsymbol{\omega}^*\|_2, \|\mathtt{A}^\mathsf{T}(\widehat{\boldsymbol{\theta}} - \boldsymbol{\theta}^*)\|_2) \leq (4/\sqrt{S})\|\boldsymbol{\omega}^* - \boldsymbol{\omega}_S^*\|_1$.

## 5.2 The noisy case

The assumption $\sigma = 0$ is an idealization of the reality that has the advantage of simplifying the mathematical derivations. While such a simplified setting is useful for conveying the main ideas behind the proposed methodology, it is of major practical importance to discuss the extensions to the more realistic *noisy* model. To this end, we introduce the vector $\widehat{\boldsymbol{\xi}}$ of estimated residuals satisfying $\mathtt{A}^{\mathsf{T}}\widehat{\boldsymbol{\theta}} = \widehat{\boldsymbol{\omega}} + \mathrm{diag}(\mathtt{C}^{\mathsf{T}}\widehat{\boldsymbol{\theta}})\,\widehat{\boldsymbol{\xi}}$ and $\|\widehat{\boldsymbol{\xi}}\|_\infty \le \sigma$.

**Theorem 3.** *Let the assumptions of Theorem 2 be fulfilled. If for some $\epsilon > 0$ we have* $\max(\|\mathrm{diag}(\mathtt{C}^{\mathsf{T}}\widehat{\boldsymbol{\theta}})\widehat{\boldsymbol{\xi}}\|_2; \|\mathrm{diag}(\mathtt{C}^{\mathsf{T}}\boldsymbol{\theta}^*)\boldsymbol{\xi}\|_2) \le \epsilon$, *then*

$$\|\widehat{\boldsymbol{\beta}} - \boldsymbol{\beta}^*\|_2 \le C_0\|\boldsymbol{\omega}^* - \boldsymbol{\omega}_S^*\|_1 + C_1\epsilon \tag{17}$$

*where $C_0$ and $C_1$ are some constants.*

*Proof.* Let us define $\boldsymbol{\eta} = \mathrm{diag}(\mathtt{C}^{\mathsf{T}}\boldsymbol{\theta}^*)\boldsymbol{\xi}$ and $\widehat{\boldsymbol{\eta}} = \mathrm{diag}(\mathtt{C}^{\mathsf{T}}\widehat{\boldsymbol{\theta}})\widehat{\boldsymbol{\xi}}$. On the one hand, in view of (15), we have $\|\mathbf{h}_{(T_0 \cup T_1)^c}\|_2 \le \|\mathbf{h}_{T_0}\|_2 + 2S^{-1/2}\|\boldsymbol{\omega}_{T_0^c}^*\|_1$ with $\mathbf{h} = \boldsymbol{\omega}^* - \widehat{\boldsymbol{\omega}}$. On the other hand, since $\mathbf{h} = \mathtt{A}^{\mathsf{T}}\mathbf{g} + \widehat{\boldsymbol{\eta}} - \boldsymbol{\eta}$, we have

$$\|\mathbf{h}_{(T_0 \cup T_1)^c}\|_2 \ge \|\mathtt{A}_{(T_0 \cup T_1)^c}^{\mathsf{T}}\mathbf{g}\|_2 - \|\widehat{\boldsymbol{\eta}}_{(T_0 \cup T_1)^c}\|_2 - \|\boldsymbol{\eta}_{(T_0 \cup T_1)^c}\|_2 \ge \|\mathtt{A}_{(T_0 \cup T_1)^c}^{\mathsf{T}}\mathbf{g}\|_2 - 2\epsilon$$

and $\|\mathbf{h}_{T_0}\|_2 \le \|\mathtt{A}_{T_0}^{\mathsf{T}}\mathbf{g}\|_2 + \|\widehat{\boldsymbol{\eta}}_{T_0}\|_2 + \|\boldsymbol{\eta}_{T_0}\|_2 \le \|\mathtt{A}_{T_0}^{\mathsf{T}}\mathbf{g}\|_2 + 2\epsilon$. These inequalities imply that

$$\|\mathtt{A}^{\mathsf{T}}\mathbf{g}\|_2 \le \|\mathtt{A}_{T_0 \cup T_1}^{\mathsf{T}}\mathbf{g}\|_2 + \|\mathtt{A}_{T_0}^{\mathsf{T}}\mathbf{g}\|_2 + 4\epsilon + 2S^{-1/2}\|\boldsymbol{\omega}_{T_0^c}^*\|_1$$

$$\le (\delta_{T_0 \cup T_1} + \delta_{T_0})\|\mathtt{A}^{\mathsf{T}}\mathbf{g}\|_2 + 4\epsilon + 2S^{-1/2}\|\boldsymbol{\omega}_{T_0^c}^*\|_1.$$

To complete the proof, it suffices to remark that

$$\|\widehat{\boldsymbol{\beta}} - \boldsymbol{\beta}^*\|_2 \le \|\mathbf{h}\|_2 + \|\mathbf{g}\|_2 \le \|\mathtt{A}^T\mathbf{g}\|_2 + \|\mathbf{g}\|_2 + 2\epsilon \le (1 + \lambda^{-1})\|\mathtt{A}^{\mathsf{T}}\mathbf{g}\|_2 + 2\epsilon$$

$$\le \frac{1 + \lambda^{-1}}{1 - \delta_{T_0 \cup T_1} - \delta_{T_0}}(4\epsilon + 2S^{-1/2}\|\boldsymbol{\omega}_{T_0^c}^*\|_1). \qquad \square$$

## 5.3 Discussion

The main assumption in Theorems 2 and 3 is that $\delta_{T_0}(\boldsymbol{\theta}^*) + \delta_{T_0 \cup T_1}(\boldsymbol{\theta}^*) < 1$. While this assumption is by no means necessary, it should be recognized that it cannot be significantly relaxed. In fact, the condition $\delta_{T_0}(\boldsymbol{\theta}^*) < 1$ is necessary for $\boldsymbol{\theta}^*$ to be consistently estimated. Indeed, if $\delta_{T_0}(\boldsymbol{\theta}^*) = 1$, then it is possible to find $\boldsymbol{\theta}' \in \partial\mathcal{P}$ such that $\mathtt{A}_{T_0^c}^{\mathsf{T}}\boldsymbol{\theta}^* = \mathtt{A}_{T_0^c}^{\mathsf{T}}\boldsymbol{\theta}'$, which makes the problem of robust estimation ill-posed, since both $\boldsymbol{\theta}^*$ and $\boldsymbol{\theta}'$ satisfy (7) with the same number of outliers.

Note also that the mapping $J \mapsto \delta_J(\boldsymbol{\theta})$ is subadditive, that is $\delta J \cup J'(\boldsymbol{\theta}) \le \delta_J(\boldsymbol{\theta}) + \delta_{J'}(\boldsymbol{\theta})$. Therefore, the condition of Thm. 2 is fulfilled as soon as $\delta_J(\boldsymbol{\theta}^*) < 1/3$ for every index set $J$ of cardinality $\le S$. Thus, the condition $\max_{J:|J| \le S} \delta_S(\boldsymbol{\theta}^*) < 1/3$ is sufficient for identifying $\boldsymbol{\theta}^*$ in presence of $S$ outliers, while $\max_{J:|J| \le S} \delta_S(\boldsymbol{\theta}^*) < 1$ is necessary.

A simple upper bound on $\delta_J$, obtained by replacing the sup over $\partial\mathcal{P}$ by the sup over $\mathbb{R}^M$, is $\delta_J(\boldsymbol{\theta}) \le \|\mathtt{O}_J^{\mathsf{T}}\|, \forall \boldsymbol{\theta} \in \partial\mathcal{P}$, where $\mathtt{O} = \mathtt{O}(\mathtt{A})$ stands for the $\mathrm{Rank}(\mathtt{A}) \times N$ matrix with orthonormal rows spanning the image of $\mathtt{A}^{\mathsf{T}}$. The matrix norm is understood as the largest singular value. Note that for a given $J$, the computation of $\|\mathtt{O}_J^{\mathsf{T}}\|$ is far easier than that of $\delta_J(\boldsymbol{\theta})$.

We emphasize that the model we have investigated comprises the robust linear model as a particular case. Indeed, if the last row of the matrix $\mathtt{A}$ is equal to zero as well as all the rows of $\mathtt{C}$ except the last row which that has all the entries equal to one, then the model described by (7) is nothing else but a linear model with unknown noise variance.

To close this section, let us stress that other approaches (cf., for instance, [9, 7, 1]) recently introduced in sparse learning and estimation may potentially be useful for the problem of robust estimation.

## 6 Numerical illustration

We implemented the algorithm in MatLab, using the SeDuMi package for solving LPs [28]. We applied our algorithm of robust estimation to the well-known dinosaur sequence [5]. which consists

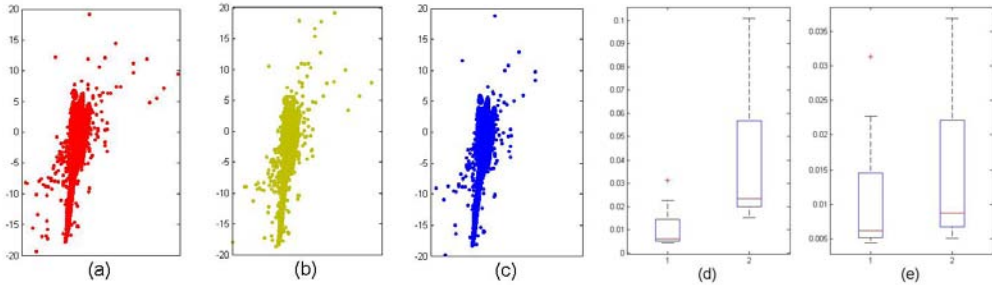

Figure 2: *(a)-(c) Overhead view of the scene points estimated by the KK-procedure (a), by the SH-procedure (b) and by our procedure. (d) Boxplots of the errors when estimating the camera centers by our procedure (left) and by the KK-procedure. (e) Boxplots of the errors when estimating the camera centers by our procedure (left) and by the SH-procedure.*

of 36 images of a dinosaur on a turntable, see Fig. 1 (a) for one example. The 2D image points which are tracked across the image sequence and the projection matrices of 36 cameras are provided as well. There are 16,432 image points corresponding to 4,983 scene points. This data is severely affected by outliers which results in a very poor accuracy of the "blind" $L_\infty$-cost minimization procedure. Its maximal RE equals 63 pixel and, as shown in Fig. 1, the estimated camera centers are not on the same plane and the scatter plot of scene points is inaccurate.

We ran our procedure with $\sigma = 0.5$ pixel. If for $p$th measurement $|\omega_p/\mathbf{c}_p^{\mathsf{T}}\boldsymbol{\theta}|$ was larger than $\sigma/4$, then the it has been considered is an outlier and removed from the dataset. The corresponding 3D scene point was also removed if, after the step of outlier removal, it was seen by only one camera. This resulted in removing $1,306$ image points and 297 scene points. The plots (d) and (e) of Fig. 1 show the estimated camera centers and estimated scene points. We see, in particular, that the camera centers are almost coplanar. Note that in this example, the second step of the procedure described in Section 4.3 does not improve on the estimator computed at the first step. Thus, an accurate estimate is obtained by solving only one linear program.

We compared our procedure with the procedures proposed by Sim and Hartley [27], hereafter referred to as SH-procedure, and by Kanade and Ke [19], hereafter KK-procedure. For the SH-procedure, we iteratively computed the $L_\infty$-cost minimizer by removing, at each step $j$, the measurements that had a RE larger than $E_{max,j} - 0.5\epsilon$, where $E_{max,j}$ was the largest RE. We have stopped the SH-procedure when the number of removed measurements exceeded 1,500. This number has been attained after 53 cycles. Therefore, the execution time was approximately 50 times larger than for our procedure. The estimator obtained by SH-procedure has a maximal RE equal to 1.33 pixel, whereas the maximal RE for our estimator is of 0.62 pixel. Concerning the KK-procedure, we run it with the parameter value $m = N - N_O = 15,000$, which is approximately the number of inliers detected by our method. Recall that the KK-procedure aims at minimizing the $m$th largest RE. As shown in Fig. 2, our procedure performs better than that of [19].

## 7  Conclusion

In this paper, we presented a rigorous Bayesian framework for the problem of translation estimation and triangulation that have leaded to a new robust estimation procedure. We have formulated the problem under consideration as a nonlinear inverse problem with a high-dimensional unknown parameter-vector. This parameter-vector encapsulates the information on the scene points and the camera locations, as well as the information on the location of outliers in the data. The proposed estimator exploits the sparse nature of the vector of outliers through $L_1$-norm minimization. We have given the mathematical proof of the result demonstrating the efficiency of the proposed estimator under mild assumptions. Real data analysis conducted on the dinosaur sequence supports our theoretical results.

## Acknowledgments

The work of the first author was partially supported by ANR under grants Callisto and Parcimonie.

## Footnotes

[1]To get a matrix of the same size as $\mathtt{A}$, in the matrix $\mathtt{C}$ each column is duplicated two times.

[2]The most common choice is $f(x) = e^{-x}$ corresponding to the multivariate Laplace density.

[3] We assume for simplicity that $\partial\mathcal{P}$ is compact.

[4]in absolute value

[5]http://www.robots.ox.ac.uk/~vgg/data1.html

# References

[1] F. Bach. Bolasso: model consistent Lasso estimation through the bootstrap. In *Twenty-fifth International Conference on Machine Learning (ICML)*, 2008. 7

[2] P. J. Bickel, Y. Ritov, and A. B. Tsybakov. Simultaneous analysis of lasso and Dantzig selector. *Ann. Statist.*, 37(4):1705–1732, 2009. 2, 6

[3] E. Candès and T. Tao. The Dantzig selector: statistical estimation when $p$ is much larger than $n$. *Ann. Statist.*, 35(6):2313–2351, 2007. 6

[4] E. J. Candès. The restricted isometry property and its implications for compressed sensing. *C. R. Math. Acad. Sci. Paris*, 346(9-10):589–592, 2008. 6

[5] E. J. Candès and P. A. Randall. Highly robust error correction by convex programming. *IEEE Trans. Inform. Theory*, 54(7):2829–2840, 2008. 2

[6] E. J. Candès, J. K. Romberg, and T. Tao. Stable signal recovery from incomplete and inaccurate measurements. *Comm. Pure Appl. Math.*, 59(8):1207–1223, 2006. 2, 6

[7] C. Chesneau and M. Hebiri. Some theoretical results on the grouped variables Lasso. *Math. Methods Statist.*, 17(4):317–326, 2008. 7

[8] A. S. Dalalyan, A. Juditsky, and V. Spokoiny. A new algorithm for estimating the effective dimension-reduction subspace. *Journal of Machine Learning Research*, 9:1647–1678, Aug. 2008. 3

[9] A. S. Dalalyan and A. B. Tsybakov. Aggregation by exponential weighting, sharp PAC-bayesian bounds and sparsity. *Machine Learning*, 72(1-2):39–61, 2008. 7

[10] D. Donoho, M. Elad, and V. Temlyakov. Stable recovery of sparse overcomplete representations in the presence of noise. *IEEE Trans. Inform. Theory*, 52(1):6–18, 2006. 2, 6

[11] D. L. Donoho and X. Huo. Uncertainty principles and ideal atomic decomposition. *IEEE Trans. Inform. Theory*, 47(7):2845–2862, 2001. 6

[12] O. Enqvist and F. Kahl. Robust optimal pose estimation. In *ECCV*, pages I: 141–153, 2008. 2

[13] R. Hartley and F. Kahl. Optimal algorithms in multiview geometry. In *ACCV*, volume 1, pages 13 – 34, Nov. 2007. 2

[14] R. Hartley and F. Kahl. Global optimization through rotation space search. *IJCV*, 2009. 2

[15] R. I. Hartley and F. Schaffalitzky. $L_\infty$ minimization in geometric reconstruction problems. In *CVPR (1)*, pages 504–509, 2004. 2, 3

[16] R. I. Hartley and A. Zisserman. *Multiple View Geometry in Computer Vision*. Cambridge University Press, June 2004. 1, 3

[17] F. Kahl. Multiple view geometry and the $L_\infty$-norm. In *ICCV*, pages 1002–1009. IEEE Computer Society, 2005. 3

[18] F. Kahl and R. I. Hartley. Multiple-view geometry under the $L_\infty$ norm. *IEEE Trans. Pattern Analysis and Machine Intelligence*, 30(9):1603–1617, sep 2008. 2, 3

[19] T. Kanade and Q. Ke. Quasiconvex optimization for robust geometric reconstruction. In *ICCV*, pages II: 986–993, 2005. 2, 8

[20] Q. Ke and T. Kanade. Uncertainty models in quasiconvex optimization for geometric reconstruction. In *CVPR*, pages I: 1199–1205, 2006. 4

[21] H. D. Li. A practical algorithm for $L_\infty$ triangulation with outliers. In *CVPR*, pages 1–8, 2007. 2

[22] D. Martinec and T. Pajdla. Robust rotation and translation estimation in multiview reconstruction. In *CVPR*, pages 1–8, 2007. 2

[23] D. Nistér. An efficient solution to the five-point relative pose problem. *IEEE Trans. Pattern Anal. Mach. Intell*, 26(6):756–777, 2004. 1

[24] C. Olsson, A. P. Eriksson, and F. Kahl. Efficient optimization for $L_\infty$ problems using pseudoconvexity. In *ICCV*, pages 1–8, 2007. 2

[25] Y. D. Seo and R. I. Hartley. A fast method to minimize $L_\infty$ error norm for geometric vision problems. In *ICCV*, pages 1–8, 2007. 2

[26] Y. D. Seo, H. J. Lee, and S. W. Lee. Sparse structures in L-infinity norm minimization for structure and motion reconstruction. In *ECCV*, pages I: 780–793, 2008. 2

[27] K. Sim and R. Hartley. Removing outliers using the $L_\infty$ norm. In *CVPR*, pages I: 485–494, 2006. 2, 8

[28] J. F. Sturm. Using SeDuMi 1.02, a MATLAB toolbox for optimization over symmetric cones. *Optim. Methods Softw.*, 11/12(1-4):625–653, 1999. 7

[29] P. Zhao and B. Yu. On model selection consistency of Lasso. *J. Mach. Learn. Res.*, 7:2541–2563, 2006. 6

